# Adaptive Neural Networks Using MOS Charge Storage

D. B. Schwartz [1], R. E. Howard and W. E. Hubbard
AT&T Bell Laboratories
Crawfords Corner Rd.
Holmdel, N.J. 07733

**Abstract**

MOS charge storage has been demonstrated as an effective method to store the weights in VLSI implementations of neural network models by several workers [2]. However, to achieve the full power of a VLSI implementation of an adaptive algorithm, the learning operation must built into the circuit . We have fabricated and tested a circuit ideal for this purpose by connecting a pair of capacitors with a CCD like structure, allowing for variable size weight changes as well as a weight decay operation. A $2.5\mu$ CMOS version achieves better than 10 bits of dynamic range in a $140\mu \times 350\mu$ area. A $1.25\mu$ chip based upon the same cell has 1104 weights on a $3.5mm \times 6.0mm$ die and is capable of peak learning rates of at least $2 \times 10^9$ weight changes per second.

## 1 Adaptive Networks

Much of the recent excitement about neural network models of computation has been driven by the prospect of new architectures for fine grained parallel computation using analog VLSI. Adaptive systems are especially good targets for analog VLSI because the adaptive process can compensate for the inaccuracy of individual devices as easily as for the variability of the signal. However, silicon VLSI does not provide us with an ideal solution for weight storage. Among the properties of an ideal storage technology for analog VLSI adaptive systems are:

- The minimum available weight change $\Delta\omega$ must be small. The simplest adaptive algorithms optimize the weights by minimizing the output error with a steepest descent search in weight space [1]. Iterative improvement algorithms such as steepest descent are based on the heuristic assumption of 'better' weights being found in the neighborhood of 'good' ones; a heuristic that fails when the granularity of the weights is not fine enough. In the worst case, the resolution required just to represent a function can grow exponentially in the dimension of the input space.

- The weights must be able to represent both positive and negative values and the changes must be easily reversible. Frequently, the weights may cycle up and down while the adaptive process is converging and millions of incremental changes during a single training session is not unreasonable. If the weights cannot easily follow all of these changes, then the learning must be done off chip.

- The parallelism of the network can be exploited to the fullest only if the mechanism controlling weight changes is simple enough to be reproduced at each weight. Ideally, the change is determined by some easily computed combination of information local to each weight and signals global to the entire system. This type of locality, which is as much a property of the algorithm as of the hardware, is necessary to keep the wiring cost associated with learning small.

- Weight decay, $\omega' = \alpha\omega$ with $\alpha < 1$ is useful although not essential. Global decay of all the weights can be used to extend their dynamic range by rescaling when the average magnitude becomes too large. Decay of randomly chosen weights can be used both to control their magnitude [2] and to help gradient searches escape from local minima.

To implement an analog storage cell with MOS VLSI the most obvious choices are non-volatile devices like floating gate and MNOS transistors, multiplying DAC's with conventional digital storage, and dynamic analog storage on MOS capacitors. Most non-volatile devices rely upon electron tunneling to change the amount of stored charge, typically requiring a large amount of circuitry to control weight changes. DAC's have already proven themselves in situations where 5 bits or less of resolution [3] [4] are sufficient, but higher resolution is prohibitively expensive in terms of area. We will show the disadvantage of MOS charge storage, its volatility, is more than outweighed by the resolution available and ease of making weight changes.

Representation of both positive and negative weights can be obtained by storing the weights $\omega_i$ differentially on a pair of capacitors in which case

$$\omega \propto V_+ - V_-.$$

Differential storage can be used to obtain some degree of rejection of leakage and can guarantee that leakage will reduce the magnitude of the weights as compared with a scheme where the weights are defined with respect to a fixed level, in which case as a weight decays it can change signs. A constant common mode voltage also eases the design constraints on the differential input multiplier used to read out the weights. An elegant way to manipulate the weights is to transfer charge from one capacitor to the other, keeping constant the total charge on the system and thus maximizing the dynamic range available from the readout circuit.

## 2   Weight Changes

Small packets of charge can easily be transferred from one capacitor to the other by exploiting charge injection, a phenomena carefully avoided by designers of switched capacitor circuits as a source of sampling error [5] [6] [7] [8] [9]. An example of a storage cell with the simplest configuration for a charge transfer system is shown in figure 1. A pair of MOS capacitors are connected by a string of narrow MOS transistors, a long one to transfer charge and two minimum length ones to isolate

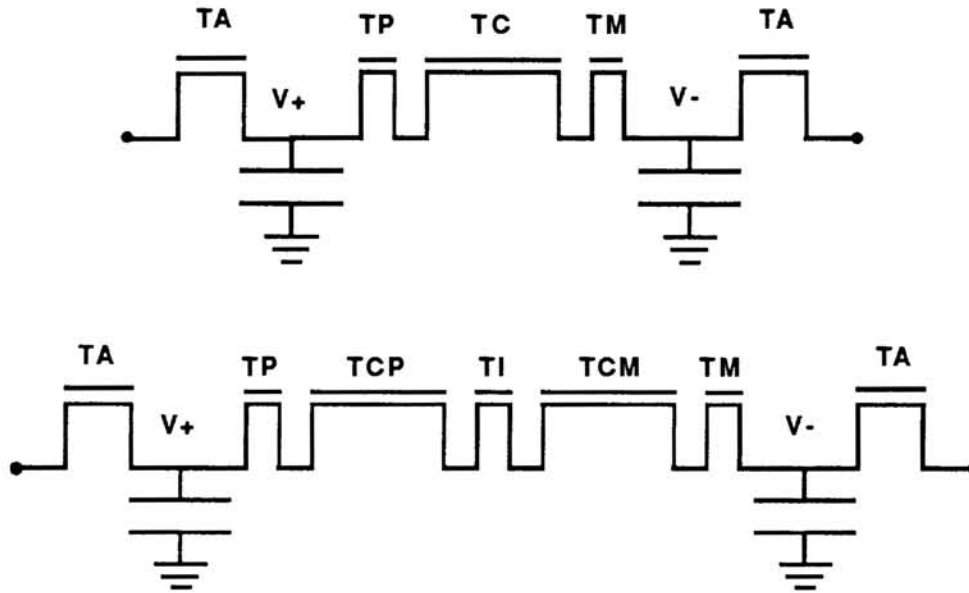

Figure 1: (a) The simplest storage cell, with provisions for only a single size increment/decrement operations and no weight decay. (b) A more sophisticated cell with facilities for weight decay. By suitable manipulation of the clock signals, the two charge transfer transistors can be used to obtain different sizes of weight changes. Both circuits are initialized by turning on the access transistors TA and charging the capacitors up to a convenient voltage, typically $V_{DD}/2$.

the charge transfer transistor from the storage nodes. For the sake of discussion, we can treat the isolation transistors as ideal switches and concentrate on the charge transfer transistor that we here assume to be an n-channel device. To increase the weight ( See figure 1 ), the charge transfer transistor (TC) and isolation transistor attached to the positive storage node (TP) are turned on. When the system has reached electrostatic equilibrium the charge transfer transistor (TC) is disconnected from the plus storage node by turning off TP and connected to the minus storage node by turning on TM. If the charge transfer transistor TC is slowly turned off, the mobile charge in its channel will diffuse into the minus node, lowering its voltage.

A detailed analysis of the charge transfer mechanism has been given elsewhere [10], but for the purpose of qualitative understanding of the circuit the inversion charge in the charge transfer transistor's channel can be approximated by

$$qN_{inv} = C_{ox}(V_G - V_{TE}).$$

where $V_{TE}$ is the effective threshold voltage and $C_{ox}$ the gate to channel capacitance of the charge transfer transistor. The effective threshold voltage is then given by

$$V_{TE} = V_{T0} + \gamma\sqrt{V_S + 2|\phi_f|}$$

where $V_{T0}$ is the threshold voltage in the absence of body effect, $\phi_f$ the fermi level, $V_S$ the source to substrate voltage, and $\gamma$ the usual body effect coefficient. An even

rougher model can be obtained by linearizing the body effect term [6]

$$\Delta Q = C_{eff}(V_G - V_T - \eta V_S)$$

where $C_{eff}$ contains both the gate oxide capacitance and the effects of parasitic capacitance and $\eta = \gamma/2\sqrt{|2\phi_f|}$ . Within the linearized approximation, the change in voltage on a storage node with capacitance $C_{store}$ after $n$ transfers is

$$V_n = V_0 + \frac{1}{\eta}(V_G - V_T - \eta V_0)(1 - \exp(-\alpha n)) \qquad (1)$$

with $\alpha = C_{eff}/C_{store}$ and where $V_0$ is the initial voltage on the storage node. Due to the dependence of the size of the transfer on the stored voltage, when the transfer direction is reversed the increment size changes unless the stored voltages on the capacitors are equal. This can be partially compensated for by using complementary pairs of p-channel and n-channel charge transfer transistors, in effect using a string of transmission gates to perform charge transfers. A weight decay operation can be introduced by using the more complex string of charge transfer transistors shown in figure 1b. A weight decay is initiated by turning off the transistor in the middle of the string (TI) and turning on all the other transistors. When the two sides of the charge transfer string have equilibrated with their respective storage nodes, the connections to the storage nodes ( TM and TP ) are turned off and the two charge transfer transistors ( TCP and TCM ) are allowed to exchange charge by turning on the transistor, TI, which separates them. When two equal charge packets have been obtained TI is turned off again and the charge packets held by TCP and TCM are injected back into the storage capacitors. The resulting change in the stored weight is

$$\Delta V^{decay} = -\frac{C_{eff}}{C_{ox}}(V_+ - V_-).$$

which corresponds to multiplying the weight by a constant $\alpha < 1$ as desired. Besides allowing for weight decay, the more complex charge string shown in figure 1b can also be used to obtain different size weight changes by using different clock sequences.

## 3  Experimental Evaluation

Test chips have been fabricated in both $1.25\mu$ and $2.5\mu$ CMOS, using the AT&T Twin Tub technology[11]. To evaluate the properties of an individual cell, especially the charge transfer mechanism, an isolated test structure consisting of five storage cells was built on one section of the $2.5\mu$ chip . The storage cells were differentially read out by two quadrant transconductance amplifiers whose input-output characteristics are shown in figure 2. By using the bias current of the amplifiers as an input, the amplifiers were used as two quadrant multipliers. Since many neural network models call for a sigmoidal nonlinearity, no attempt was made to linearize the operation of the multiplier. The output currents of the five multipliers were summed by a single output wire and the voltages on each of the ten capacitors were

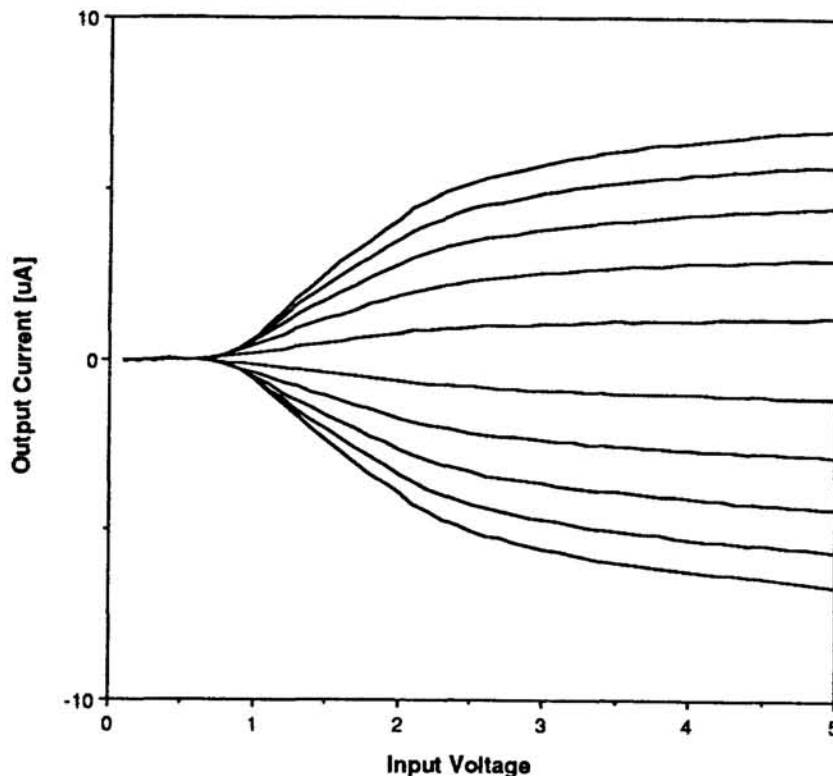

Figure 2: A family of transfer characteristics from one of the transconductance multipliers for several different values of stored weight. The different branches of the curves are each separated by ten large charge transfers. No attempt was made to linearize the input/output characteristic since many neural network models call for non-linearities.

buffered by voltage followers to allow for detailed examination of the inner workings of the cell.

After trading off between hold time, resolution and area we decided upon $20\mu$ long charge transfer transistors and $2000\mu^2$ storage capacitors with $2.5\mu$ technology based upon the minimum channel width of $2.5\mu$. For a $20\mu$ long channel and a $2.5V$ gate to source voltage the channel transit time $\tau_o$ is approximately $5\,ns$ and charge transfer clock frequencies exceeding $10MHz$ are possible without measurable pumping of charge into the substrate. The $2.5\mu$ wide access transistors were $12\mu$ long, leading to leakage rates from the individual capacitors of about 1% of the stored value in $100s$, limited by surface leakage in our unpassivated test structures. Even with uncapped wafers, the leakage was small enough to allow all the tests described here to be made without special provisions for environmental control of either temperature or humidity. As mentioned earlier, the more complex set of charge transfer transistors needed to introduce weight decay can also be used to obtain several different size of charge transfers, a small weight change by using the two long transistors in sequence and a coarse one by treating the two long transistors and the isolation transistor separating them as a single device. Using the small weight changes, the worst case resolution was 10 bits ( near $\Delta V = 0$ ) and the results where in excellent agreement with the predictions of equation 1

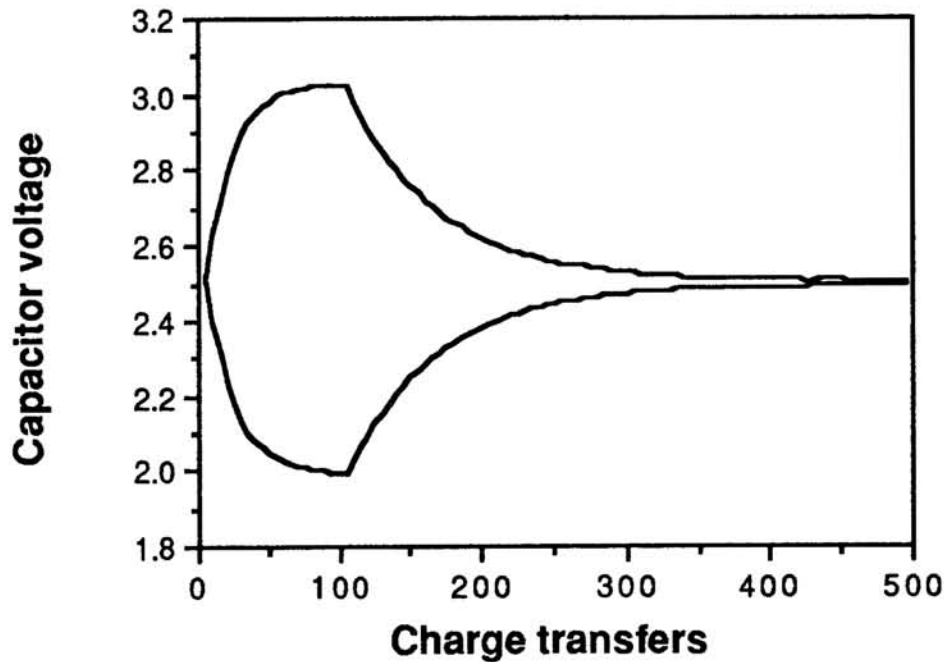

Figure 3: The voltage on the two storage capacitors when the weight is initially set to saturation using large increments and then reduced back towards zero using weight decay. The granularity of the curves is an experimental artifact of the digital voltmeter's resolution.

using the effective capacitance as a fitting parameter. In the figure 3 we use large charge transfers to quickly increment the weight up to its maximum value and then reduce it back to zero with weight decays, demonstrating the expected exponential dependence of the stored voltage on the number of weight decays. Even under repeated cycling up and down through the entire differential voltage range of the cell, the total amount of charge on the cell remained constant for frequencies under $10MHz$ with the exception of the expected losses due to leakage.

The long term goal of this work is to develop analog VLSI chips that are complete 'learning machines', capable of modified their own weights when provided with input data and some feedback based on the output of the network. However, the study of learning algorithms is in a state of flux and few, if any, algorithms have been optimized for VLSI implementation. Rather than cast an inappropriate algorithm in silicon, we have designed our first chips to be used as adaptive systems with an external controller, allowing us to develop algorithms that are appropriate for the medium once we understand its properties. The networks are organized as rectangular matrix multipliers with voltage inputs and current outputs with 46 inputs and 24 outputs in a 96 pin package for the $1.25\mu$ chip. Since none of the analog input/output lines of the chip are multiplexed, larger and more complicated networks can be built by cascading several chips.

To the digital controller, the chip looks like a 1104 × 2 static RAM with some extra clock inputs to drive the charge transfers. The charge transfer clock signals are

distributed globally and are connected to the individual strings of charge transfer transistors through a pair of $2 \times 2$ cross bar switches controlled by two bits of static RAM local to each cell . The use of a pair of cross bar switches is necessitated by the faciltities for weight decay; if the simpler charge transfer string shown in figure 1a were used then only a single switch would be needed. When both a cell's RAMs are zeroed, the global charge transfer lines are not connected to the charge transfer transistors. The global lines are connected to the individual strings of charge transfer transistors either normally or in reverse depending upon which RAM cell contains a one. By reversing the order of the signals on the charge transfer lines, a weight change can also be reversed. Neglecting the dependence of the size of the charge transfer upon stored weight, the RAM's represent a weight change vector $\Delta \vec{\omega}_{ij}$ with components $\Delta \omega_{ij} \in [-1, 0, 1]$. Once a weight change vector has been written serially to the RAM's, the weight changes along that vector are made in parallel by manipulating the charge transfer lines. This architecture is also a powerful way to implement programable networks of fixed weights since an arbitrary matrix of 10 bit weights can be written to the chip in a few milliseconds or less if an efficient decomposition of the desired weight vector into global charge transfers is made. In view of the speed with which the chip can evaluate the output of a network, an overhead of less than a percent for a refresh operation is acceptable in many applications.

# 4  Conclusions

We have implemented a generic chip to facilitate studying adaptive networks by building them in analog VLSI. By exploiting the well known properties of charge storage and charge injection in a novel way, we have achieved a high enough level of complexity ( $> 10^3$ weights and 10 bits of analog depth) to be interesting, in spite of the limitation to a modest $6.00mm \times 3.5mm$ die size required by a multi-project fabrication run. If the cell were optimized to represent fixed weight networks by eliminating weight decay and bi-directional weight changes, the density could easily be increased by a factor of two with no loss in resolution. Once a weight change vector has been written to the RAM cells, charge transfers can be clocked at a rate of $2MHz$ chip corresponds to a peak learning rate of $2 \times 10^9$ updates/second, exceeding the speeds of 'digital neurocomputers' based upon DSP chips by two orders of magnitude.

## Acknowledgements
A large group of people assisted the authors in taking this work from concept to silicon, a few of whom we single out for mention here. The IDA design tools used for the layouts were provided and supported by D. D. Hill and D. D. Shugard at Murray Hill and the $1.25\mu$ process was supported by D. Wroge and R. Ashton. The first author wishes to acknowledge helpful discussions with H. P. Graf, S. Mackie and G. Taylor, with special thanks to R. G. Swartz.

## Footnotes

[1]Now at GTE Laboratories, 40 Sylvan Rd., Waltham, Mass 02254 dbs@gte.com%relay.cs.net

[2]For example, see the papers by Mann and Gilbert, Walker and Akers, and Murray *et. al.* in this proceedings

# References

[1] Bernard Widrow and Samuel D. Stearns. *Adaptive Signal Processing.* Prentice-Hall, Inc., Englewood Cliffs, N. J., 1985.

[2] D. H. Ackley, G. E. Hinton, and T. J. Sejnowski. A learning algorithm for Boltzman machines. *Cognitive Science*, 9:147, 1985.

[3] Jack Raffel, James Mann, Robert Berger, Antonio Soares, and Sheldon Gilbert. A generic architecture for wafer-scale neuromorphic systems. In *IEEE First International Conference on Neural Networks. Volume III*, page 501, 1987.

[4] Joshua Alspector, Bhusan Gupta, and Robert B. Allen. Performance of a stochastic learning microchip. In *Advances in Neural Network Information Processing Systems*, 1988.

[5] William B. Wilson, Hisham Z. Massoud, Eric J. Swanson, Rhett T. George, and Richard B. Fair. Measurement and modeling of charge feedthrough in n-channel MOS analog switches. *IEEE Journal of Solid-State Circuits*, SC-20(6):1206–1213, 1985.

[6] George Wegmann, Eric A. Vittoz, and Fouad Rahali. Charge injection in analog MOS switches. *IEEE Journal of Solid-State Circuits*, SC-20(6):1091–1097, 1987.

[7] James A. Kuo, Robert W. Dutton, and Bruce A. Wooley. MOS pass transistor turn-off transient analysis. *IEEE Transactions on Electron Devices*, ED-33(10):1545–1555, 1986.

[8] James R. Kuo, Robert W. Dutton, and Bruce A. Wooley. Turn-off transients in circular geometry MOS pass transistors. *IEEE Journal Solid-State Circuits*, SC-21(5):837–844, 1986.

[9] Je-Hurn Shieh, Mahesh Patil, and Bing J. Sheu. Measurement and analysis of charge injection in MOS analog switches. *IEEE Journal of Solid State Circuits*, SC-22(2):277–281, 1987.

[10] R. E. Howard, D. B. Schwartz, and W. E. Hubbard. A programmable analog neural network chip. *IEEE Journal of Solid-State Circuits*, 24, 1989.

[11] J. Argraz-Guerena, R. A. Ashton, W. J. Bertram, R. C. Melin, R. C. Sun, and J. T. Clemens. Twin Tub III - A third generation CMOS. In *Proceedings of the International Electron Device Meeting*, 1984. Citation P63-6.
